# Evaluating Search Engines by Modeling the Relationship Between Relevance and Clicks

**Ben Carterette**[*]
Center for Intelligent Information Retrieval
University of Massachusetts Amherst
Amherst, MA 01003
carteret@cs.umass.edu

**Rosie Jones**
Yahoo! Research
3333 Empire Ave
Burbank, CA 91504
jonesr@yahoo-inc.com

## Abstract

We propose a model that leverages the millions of clicks received by web search engines to predict document relevance. This allows the comparison of ranking functions when clicks are available but complete relevance judgments are not. After an initial training phase using a set of relevance judgments paired with click data, we show that our model can predict the relevance score of documents that have not been judged. These predictions can be used to evaluate the performance of a search engine, using our novel formalization of the confidence of the standard evaluation metric discounted cumulative gain (DCG), so comparisons can be made across time and datasets. This contrasts with previous methods which can provide only pair-wise relevance judgments between results shown for the same query. When no relevance judgments are available, we can identify the better of two ranked lists up to 82% of the time, and with only two relevance judgments for each query, we can identify the better ranking up to 94% of the time. While our experiments are on sponsored search results, which is the financial backbone of web search, our method is general enough to be applicable to algorithmic web search results as well. Furthermore, we give an algorithm to guide the selection of additional documents to judge to improve confidence.

## 1 Introduction

Web search engine evaluation is an expensive process: it requires *relevance judgments* that indicate the degree of relevance of each document retrieved for each query in a testing set. In addition, reusing old relevance judgements to evaluate an updated ranking function can be problematic, since documents disappear or become obsolete, and the distribution of queries entered changes [15]. Click data from web searchers, used in aggregate, can provide valuable evidence about the relevance of each document. The general problem with using clicks as relevance judgments is that clicks are biased. They are biased to the top of the ranking [12], to trusted sites, to attractive abstracts; they are also biased by the type of query and by other things shown on the results page. To cope with this, we introduce a family of models relating clicks to relevance. By conditioning on clicks, we can predict the relevance of a document or a set of documents.

Joachims et al. [12] used eye-tracking devices to track what documents users looked at before clicking. They found that users tend to look at results ranked higher than the one they click on more often than they look at results ranked lower, and this information can in principle be used to train a search engine using these "preference judgments"[10]. The problem with using preference judgments inferred from clicks for learning is that they will tend to learn to reverse the list. A click at the lowest rank is preferred to everything else, while a click at the highest rank is preferred to nothing

---

[*]Work done while author was at Yahoo!

else. Radlinski and Joachims [13] suggest an antidote to this: randomly swapping adjacent pairs of documents. This ensures that users will not prefer document $i$ to document $i + 1$ solely because of rank. However, we may not wish to show a suboptimal document ordering in order acquire data.

Our approach instead will be to use *discounted cumulative gain* ($DCG$ [9]), an evaluation metric commonly used in search engine evaluation. Using click data, we can estimate the *confidence* that a difference in $DCG$ exists between two rankings without having any relevance judgments for the documents ranked. We will show how a comparison of ranking functions can be performed when clicks are available but complete relevance judgments are not. After an initial training phase with a few relevance judgments, the relevance of unjudged documents can be predicted from clickthrough rates. The confidence in the evaluation can be estimated with the knowledge of which documents are most frequently clicked. Confidence can be dramatically increased with only a few more judiciously chosen relevance judgments.

Our contributions are (1) a formalization of the information retrieval metric DCG as a random variable (2) analysis of the sign of the difference between two DCGs as an indication that one ranking is better than another (3) empirical demonstration that combining click-through rates over all results on the page is better at predicting the relevance of the document at position $i$ than just the click-through rate at position $i$ (4) empirically modeling relevance of documents using clicks, and using this model to estimate DCG (5) empirical evaluation of comparison of different rankings using DCG derived from clicks (6) an algorithm for selection of minimal numbers of documents for manual relevance judgement to improve the confidence in DCG over the estimate derived from clicks alone.

Section 2 covers previous work on using clickthrough rates and on estimating evaluation metrics. Section 3 describes the evaluation of web retrieval systems using the metric *discounted cumulative gain* (DCG) and shows how to estimate the confidence that a difference exists when relevance judgments are missing. Our model for predicting relevance from clicks is described in Section 4. We discuss our data in Section 5 and in Section 6 we return to the task of estimating relevance for the evaluation of search engines. Our experiments are conducted in the context of sponsored search, but the methods we use are general enough to translate to general web search engines.

## 2   Previous Work

There has been a great deal of work on low-cost evaluation in TREC-type settings ([20, 6, 16, 5] are a few), but we are aware of little for the web. As discussed above, Joachims [10, 12] and Radlinski and Joachims [13] conducted seminal work on using clicks to infer user preferences between documents. Agichtein et al.[2, 1] used and applied models of user interaction to predict preference relationships and to improve ranking functions. They use many features beyond clickthrough rate, and show that they can learn preference relationships using these features. Our work is superficially similar, but we explicitly model dependencies among clicks for results at different ranks with the purpose of learning probabilistic relevance judgments. These relevance judgments are a stronger result than preference ordering, since preference ordering can be derived from them. In addition, given a strong probabilistic model of relevance from clicks, better combined models can be built.

Dupret et al. [7] give a theoretical model for the rank-position effects of click-through rate, and build theoretical models for search engine quality using them. They do not evaluate estimates of document quality, while we empirically compare relevance estimated from clicks to manual relevance judgments. Joachims [11] investigated the use of clickthrough rates for evaluation, showing that relative differences in performance could be measured by interleaving results from two ranking functions, then observing which function produced results that are more frequently clicked. As we will show, interleaving results can change user behavior, and not necessarily in a way that will lead to the user clicking more relevant documents.

Soboroff [15] proposed methods for maintaining the relevance judgments in a corpus that is constantly changing. Aslam et al. [3] investigated minimum variance unbiased estimators of system performance, and Carterette et al. [5] introduced the idea of treating an evaluation measure as a *random variable* with a distribution over all possible relevance judgments. This can be used to create an optimal sampling strategy to obtain judgments, and to estimate the *confidence* in an evaluation measure. We extend their methods to DCG.

# 3 Evaluating Search Engines

Search results are typically evaluated using Discounted Cumulative Gain (DCG) [9]. DCG is defined as the sum of the "gain" of presenting a particular document times a "discount" of presenting it at a particular rank, up to some maximum rank $\ell$: $DCG_\ell = \sum_{i=1}^{\ell} gain_i discount_i$. For web search, "gain" is typically a relevance score determined from a human labeling, and "discount" is the reciprocal of the log of the rank, so that putting a document with a high relevance score at a low rank results in a much lower discounted gain than putting the same document at a high rank.

$$DCG_\ell = rel_1 + \sum_{i=2}^{\ell} \frac{rel_i}{\log_2 i}$$

The constants $rel_i$ are the relevance scores. Human assessors typically judge documents on an ordinal scale, with labels such as "Perfect", "Excellent", "Good", "Fair", and "Bad". These are then mapped to a numeric scale for use in DCG computation. We will denote five levels of relevance $a_j$, with $a_1 > a_2 > a_3 > a_4 > a_5$. In this section we will show that we can compare ranking functions *without* having labeled all the documents.

## 3.1 Estimating DCG from Incomplete Information

DCG requires that the ranked documents have been judged with respect to a query. If the index has recently been updated, or a new algorithm is retrieving new results, we have documents that have not been judged. Rather than ask a human assessor for a judgment, we may be able to infer something about DCG based on the judgments we already have.

Let $X_i$ be a random variable representing the relevance of document $i$. Since relevance is ordinal, the distribution of $X_i$ is multinomial. We will define $p_{ij} = p(X_i = a_j)$ for $1 \le j \le 5$ with $\sum_{j=1}^{5} p_{ij} = 1$. The expectation of $X_i$ is $E[X_i] = \sum_{j=1}^{5} p_{ij} a_j$, and its variance is $Var[X_i] = \sum_{j=1}^{5} p_{ij} a_j^2 - E[X_i]^2$.

We can then express $DCG$ as a random variable:

$$DCG_\ell = X_1 + \sum_{i=2}^{\ell} \frac{X_i}{\log_2 i}$$

Its expectation and variance are:

$$E[DCG_\ell] = E[X_1] + \sum_{i=2}^{\ell} \frac{E[X_i]}{\log_2 i} \tag{1}$$

$$Var[DCG_\ell] = Var[X_1] + \sum_{i=2}^{\ell} \frac{Var[X_i]}{(\log_2 i)^2} + 2 \sum_{i=1}^{\ell} \frac{Cov(X_1, X_i)}{\log_2 i} + 2 \sum_{1 < i < j} \frac{Cov(X_i, X_j)}{\log_2 i \cdot \log_2 j} - E[DCG_\ell]^2 \tag{2}$$

If the relevance of documents $i$ and $j$ are independent, the covariance $Cov(X_i, X_j)$ is zero.

When some relevance judgments are not available, Eq. (1) and (2) can be used to estimate confidence intervals for DCG. Thus we can compare ranking functions without having judged all the documents.

## 3.2 Comparative Evaluation

If we only care about whether one index or ranking function outperforms another, the actual values of DCG matter less than the sign of their difference. We now turn our attention to estimating the sign of the difference with high confidence. We redefine $DCG$ in terms of an arbitrary indexing of documents, instead of the indexing by rank we used in the previous section. Let $r_j(i)$ be the rank at which document $i$ was retrieved by system $j$. We define the *discounted gain* $g_{ij}$ of document $i$ to the DCG of system $j$ as $g_{ij} = rel_i$ if $r_j(i) = 1$, $g_{ij} = \frac{rel_i}{\log_2 r_j(i)}$ if $1 < r_j(i) \le \ell$, and $g_{ij} = 0$ if

document $i$ was not ranked by system $j$. Then we can write the difference in $DCG$ for systems 1 and 2 as

$$\Delta DCG_\ell = DCG_{\ell 1} - DCG_{\ell 2} = \sum_{i=1}^{N} g_{i1} - g_{i2} \qquad (3)$$

where $N$ is the number of documents in the entire collection. In practice we need only consider those documents returned in the top $\ell$ by either of the two systems. We can define a random variable $G_{ij}$ by replacing $rel_i$ with $X_i$ in $g_{ij}$; we can then compute the expectation of $\Delta DCG$:

$$E[\Delta DCG_\ell] = \sum_{i=1}^{N} E[G_{i1}] - E[G_{i2}]$$

We can compute its variance as well, which is omitted here due to space constraints.

### 3.3 Confidence in a Difference in DCG

Following Carterette et al. [5], we define the *confidence* in a difference in $DCG$ as the probability that $\Delta DCG = DCG_1 - DCG_2$ is less than zero. If $P(\Delta DCG < 0) \geq 0.95$, we say that we have 95% confidence that system 1 is worse than system 2: over all possible judgments that could be made to the unjudged documents, 95% of them will result in $\Delta DCG < 0$.

To compute this probability, we must consider the distribution of $\Delta DCG$. For web search, we are typically most interested in performance in the top 10 retrieved. Ten documents is too few for any convergence results, so instead we will estimate the confidence using Monte Carlo simulation. We simply draw relevance scores for the unjudged documents according to the multinomial distribution $p(X_i)$ and calculate $\Delta DCG$ using those scores. After $T$ trials, the probability that $\Delta DCG$ is less than 0 is simply the number of times $\Delta DCG$ was computed to be less than 0 divided by $T$.

How can we estimate the distribution $p(X_i)$? In the absence of any other information, we may assume it to be uniform over all five relevance labels. Relevance labels that have been made in the past provide a useful prior distribution. As we shall see below, clicks are a useful source of information that we can leverage to estimate this distribution.

### 3.4 Selecting Documents to Judge

If confidence estimates are low, we may want to obtain more relevance judgments to improve it. In order to do as little work as necessary, we should select the documents that are likely to tell us a lot about $\Delta DCG$ and therefore tell us a lot about confidence. The most informative document is the one that would have the greatest effect on $\Delta DCG$. Since $\Delta DCG$ is linear, it is quite easy to determine which document should be judged next. Eq. (3) tells us to simply choose the document $i$ that is unjudged and has maximum $|E[G_{i1}] - E[G_{i2}]|$. Algorithm 1 shows how relevance judgments would be acquired iteratively until confidence is sufficiently high. This algorithm is provably optimal in the sense that after $k$ judgments, we know more about the difference in DCG than we would with any other $k$ judgments.

---

**Algorithm 1** Iteratively select documents to judge until we have high confidence in $\Delta DCG$.

---

1: **while** $1 - \alpha \leq P(\Delta DCG < 0) \leq \alpha$ **do**
2:    $i^* \leftarrow \max_i |E[G_{i1}] - E[G_{i2}]|$ for all unjudged documents $i$
3:    judge document $i^*$
    (human annotator provides $rel_{i^*}$)
4:    $P(X_{i^*} = rel_{i^*}) \leftarrow 1$
5:    $P(X_{i^*} \neq rel_{i^*}) \leftarrow 0$
6:    estimate $P(\Delta DCG)$ using Monte Carlo simulation
7: **end while**

---

## 4 Modeling Clicks and Relevance

Our goal is to model the relationship between clicks and relevance in a way that will allow us to estimate a distribution of relevance $p(X_i)$ from the clicks on document $i$ and on surrounding

documents. We first introduce a joint probability distribution including the query $q$, the relevance $X_i$ of each document retrieved (where $i$ indicates the rank), and their respective clickthrough rates $c_i$:

$$p(q, X_1, X_2, ..., X_\ell, c_1, c_2, ..., c_\ell) = P(q, \mathbf{X}, \mathbf{c}) \tag{4}$$

Boldface $\mathbf{X}$ and $\mathbf{c}$ indicate vectors of length $\ell$.

Suppose we have a query for which we have few or no relevance judgments (perhaps because it has only recently begun to appear in the logs, or because it reflects a trend for which new documents are rapidly being indexed). We can nevertheless obtain click-through data. We are therefore interested in the conditional probability $p(\mathbf{X}|q, \mathbf{c})$.

Note that $\mathbf{X} = \{X_1, X_2, \cdots\}$ is a vector of discrete ordinal variables; doing inference in this model is not easy. To simplify, we make the assumption that the relevance of document $i$ and document $j$ are conditionally independent given the query and the clickthrough rates:

$$p(\mathbf{X}|q, \mathbf{c}) = \prod_{i=1}^{\ell} p(X_i|q, \mathbf{c}) \tag{5}$$

This gives us a separate model for each rank, while still conditioning the relevance at rank $i$ on the clickthrough rates at all of the ranks. We do not lose the dependence between relevance at each rank and clickthrough rates on other ranks. We will see the importance of this empirically in section 6.

The independence assumption allows us to model $p(X_i)$ using ordinal regression. Ordinal regression is a generalization of logistic regression to a variable with two or more outcomes that are ranked by preference.

The proportional odds model for our ordinal response variable is

$$\log \frac{p(X > a_j|q, \mathbf{c})}{p(X \le a_j|q, \mathbf{c})} = \alpha_j + \beta q + \sum_{i=1}^{\ell} \beta_i c_i + \sum_{i<k}^{\ell} \beta_{ik} c_i c_k$$

where $a_j$ is one of the five relevance levels. The sums are over all ranks in the list; this models the dependence of the relevance of the document to the clickthrough rates of everything else that was retrieved, as well as any multiplicative dependence between the clickthrough rates at any two ranks.

After the model is trained, we can obtain $p(X \le a_j|q, \mathbf{c})$ using the inverse logit function. Then $p(X = a_j|q, \mathbf{c}) = p(X \le a_j|q, \mathbf{c}) - p(X \le a_{j-1}|q, \mathbf{c})$.

A generalization to the proportional odds model is the vector generalized additive model (VGAM) described by Yee and Wild [19]. VGAM has the same relationship to ordinal regression that GAM [8] has to logistic regression. It is useful in our case because clicks do not necessarily have linear relationships to relevance. VGAM is implemented in the `R` library `VGAM`. Once the model is trained, we have $p(X = a_j)$ using the same arithmetic as for the proportional odds model.

## 5  Data

We obtained data from Yahoo! sponsored search logs for April 2006. Although we limited our data to advertisements, there is no reason in principle our method should not be applicable to general web search, since we see the same effects of bias towards the top of search results, to trusted sites and so on. We have a total of 28,961 relevance judgments for 2,021 queries. The queries are a random sample of all queries entered in late 2005 and early 2006. Relevance judgments are based on details of the advertisement, such as title, summary, and URL.

We filtered out queries for which we had no relevance judgments. We then aggregated records into distinct lists of advertisements for a query as follows: Each record $L$ consisted of a query, a search identification string, a set of advertisement ids, and for each advertisement id, the rank the advertisement appeared at and the number of times it was clicked. Different sets of results for a query, or results shown in a different order, were treated as distinct lists. We aggregated distinct lists of results to obtain a clickthrough rate at each rank for a given list of results for a given query. The clickthrough rate on each ad is simply the number of times it was clicked when served as part of list $L$ divided by the impressions, the number of times $L$ was shown to any user. We did not adjust for impression bias.

### 5.1 Dependence of Clicks on Entire Result List

Our model takes into account the clicks at *all ranks* to estimate the relevance of the document at position $i$. As the figure to the right shows, when there is an "Excellent" document at rank 1, its clickthrough rate varies depending on the relevance of the document at rank 2. For example, a "Perfect" document at rank 2 may decrease the likelihood of a click on the "Excellent" document at rank 1, while a "Fair" document at rank 2 may *increase* the clickthrough rate for rank 1. Clickthrough rate at rank 1 more than doubles as the relevance of the document at rank 2 drops from "Perfect" to "Fair".

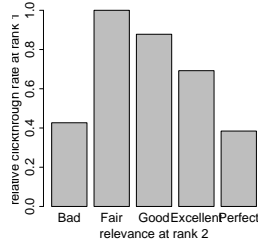

## 6 Experiments

### 6.1 Fit of Document Relevance Model

We first want to test our proposed model (Eq. (5)) for predicting relevance from clicks. If the model fits well, the distributions of relevance it produces should compare favorably to the actual relevance of the documents. We will compare it to a simpler model that does not take into account the click dependence. The two models are contrasted below:

$$\text{dependence model:} \quad p(\mathbf{X}|q, \mathbf{c}) = \prod p(X_i|q, \mathbf{c})$$

$$\text{independence model:} \quad p(\mathbf{X}|q, \mathbf{c}) = \prod p(X_i|q, c_i)$$

The latter models the relevance being conditional only on the query and its own clickthrough rate, ignoring the clickthrough rates of the other items on the page. Essentially, it discretizes clicks into relevance label bins at each rank using the query as an aid.

We removed all instances for which we had fewer than 500 impressions, then performed 10-fold cross-validation. For simplicity, the query $q$ is modeled as the aggregate clickthrough rate over all results ever returned for that query. Both models produce a multinomial distribution for the probability of relevance of a document $p(X_i)$. Predicted relevance is the expected value of this distribution: $E[X_i] = \sum_{j=1}^{5} p(X_i = a_j)a_j$.

The correlation between predicted relevance and actual relevance starts from $0.754$ at rank 1 and trends downward as we move down the list; by rank 5 it has fallen to $0.527$. Lower ranks are clicked less often; there are fewer clicks to provide evidence for relevance. Correlations for the independence model are significantly lower at each point.

Figure 1 depicts boxplots for each value of relevance for both models. Each box represents the distribution of predictions for the true value on the $x$ axis. The center line is the median prediction; the edges are the 25% and 75% quantiles. The whiskers are roughly a 95% confidence interval, with the points outside being outliers. When dependence is modeled (Figure 1(a)), the distributions are much more clearly separated from each other, as shown by the fact that there is little overlap in the boxes. The correlation between predicted and acutal relevance is $18\%$ higher, a statistically significant difference.

### 6.2 Estimating DCG

Since our model works fairly well, we now turn our attention to using relevance predictions to estimate DCG for the evaluation of search engines. Recall that we are interested in comparative evaluation—determining the sign of the difference in DCG rather than its magnitude. Our confidence in the sign is $P(\Delta DCG < 0)$, which is estimated using the simulation procedure described in Section 3.3. The simulation samples from the multinomial distributions $p(X_i)$.

**Methodology**: To be able to calculate the exact $DCG$ to evaluate our models, we need all ads in a list to have a relevance judgment. Therefore our test set will consist of all of the lists for which we have complete relevance judgments and at least $500$ impressions. The remainder will be used for training. The size of the test set is 1720 distinct lists. The training sets will include all lists for which we have at least 200 impressions, over 5000 lists. After training the model, we

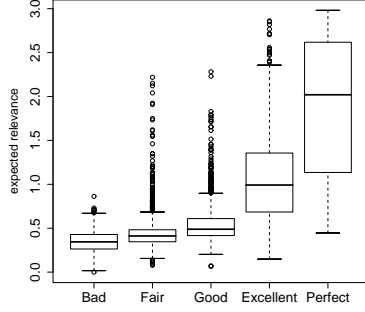
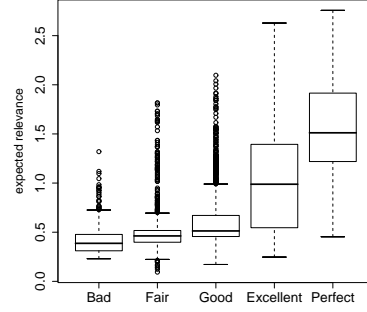

(a) Dependence model; $\rho = 0.754$          (b) No dependence modeled; $\rho = 0.638$

Figure 1: Predicted vs. actual relevance for rank 1. Correlation increases $18\%$ when dependence of relevance of the document at rank 1 on clickthrough at all ranks is modeled.

| Confidence | $0.5 - 0.6$ | $0.6 - 0.7$ | $0.7 - 0.8$ | $0.8 - 0.9$ | $0.9 - 0.95$ | $0.95 - 1.0$ |
|---|---|---|---|---|---|---|
| Accuracy clicks-only | 0.522 | 0.617 | 0.734 | 0.818 | – | – |
| Accuracy 2 judgments | 0.572 | 0.678 | 0.697 | 0.890 | 0.918 | 0.940 |

Table 1: Confidence vs. accuracy of predicting the better ranking for pairs of ranked lists using the relevance predictions of our model based on clicks alone, and with two additional judgments for each pair of lists. Confidence estimates are good predictions of accuracy.

predict relevance for the ads in the test set. We then use these expected relevances to calculate the expectation $E[DCG]$. We will compare these expectations to the true $DCG$ calculated using the actual relevance judgments. As a baseline for automatic evaluation, we will compare to the average clickthrough rate on the list $E[CTR] = \frac{1}{k} \sum c_i$, the naive approach described in our introduction. We then estimate the confidence $P(\Delta DCG < 0)$ for pairs of ranked lists for the same query and compare it to the actual percentage of pairs that had $\Delta DCG < 0$. Confidence should be less than or equal to this percentage; if it is, we can "trust" it in some sense.

**Results**: We first looked at the ability of $E[DCG]$ to predict $DCG$, as well as the ability of the average clickthrough rate $E[CTR]$ to predict $DCG$. The correlation between the latter two is $0.622$, while the correlation between the former two is $0.876$. This means we can approximate DCG better using our model than just using the mean clickthrough rate as a predictor.

The figure to the right shows actual vs. predicted relevance for ads in the test set. (This is slightly different from Figure 1: the earlier figure shows predicted results for *all* data from cross-validation while this one only shows predicted results on our test data.) The separation of the boxes shows that our model is doing quite well on the testing data, at least for rank 1. Performance degrades quite a bit as rank increases (not shown), but it is important to note that the upper ranks have the greatest effect on $DCG$—so getting those right is most important.

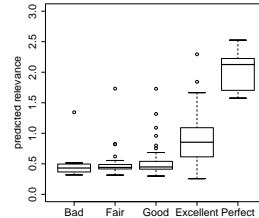

In Table 1, we have binned pairs of ranked lists by their estimated confidence. We computed the accuracy of our predictions (the percent of pairs for which the difference in $DCG$ was correctly identified) for each bin. The first line shows results when evaluating with no additional relevance judgments beyond those used for training the model: although confidence estimates tend to be low, they are accurate in the sense that a confidence estimate predicts how well we were able to distinguish between the two lists. This means that the confidence estimates provide a guide for identifying which evaluations require "hole-filling" (additional judgments).

The second line shows how results improve when only two judgments are made. Confidence estimates increase a great deal (to a mean of over $0.8$ from a mean of $0.6$), and the accuracy of the confidence estimates is not affected.

In general, performance is very good: using only the predictions of our model based on clicks, we have a very good sense of the confidence we should have in our evaluation. Judging only two more documents dramatically improves our confidence: there are many more pairs in high-confidence bins after two judgments.

## 7   Conclusion

We have shown how to compare ranking functions using expected DCG. After a single initial training phase, ranking functions can be compared by predicting relevance from clickthrough rates. Estimates of confidence can be computed; the confidence gives a lower bound on how accurately we have predicted that a difference exists. With just a few additional relevance judgments chosen cleverly, we significantly increase our success at predicting whether a difference exists. Using our method, the cost of acquiring relevance judgments for web search evaluation is dramatically reduced, when we have access to click data.

## References

[1] E. Agichtein, E. Brill, and S. T. Dumais. Improving web search ranking by incorporating user behavior information. In *Proceedings SIGIR*, pages 19–26, 2006.

[2] E. Agichtein, E. Brill, S. T. Dumais, and R. Ragno. Learning user interaction models for predicting web search result preferences. In *Proceedings SIGIR*, pages 3–10, 2006.

[3] J. A. Aslam, V. Pavlu, and E. Yilmaz. A sampling technique for efficiently estimating measures of query retrieval performance using incomplete judgments. In *Proceedings of the 22nd ICML Workshop on Learning with Partially Classified Training Data*, pages 57–66, 2005.

[4] A. Broder. A taxonomy of web search. *SIGIR Forum*, 36(2):3–10, 2002.

[5] B. Carterette, J. Allan, and R. K. Sitaraman. Minimal test collections for retrieval evaluation. In *Proceedings of SIGIR*, pages 268–275, 2006.

[6] G. V. Cormack, C. R. Palmer, and C. L. Clarke. Efficient Construction of Large Test Collections. In *Proceedings of SIGIR*, pages 282–289, 1998.

[7] G. Dupret, B. Piwowarski, C. Hurtado, and M. Mendoza. A statistical model of query log generation. In *SPIRE*, LNCS 4209, pages 217–228. Springer, 2006.

[8] T. Hastie and R. Tibshirani. Generalized additive models. *Statistical Science*, 1:297–318, 1986.

[9] K. Jarvelin and J. Kekalainen. Cumulated gain-based evaluation of ir techniques. *ACM Trans. Inf. Syst.*, 20(4):422–446, 2002.

[10] T. Joachims. Optimizing search engines using clickthrough data. In *Proceedings of KDD*, pages 133–142, 2002.

[11] T. Joachims. Evaluating retrieval performance using clickthrough data. In *Text Mining*, pages 79–96. 2003.

[12] T. Joachims, L. A. Granka, B. Pan, H. Hembrooke, and G. Gay. Accurately interpreting clickthrough data as implicit feedback. In *Proceedings of SIGIR*, pages 154–161, 2005.

[13] F. Radlinski and T. Joachims. Minimally invasive randomization fro collecting unbiased preferences from clickthrough logs. In *Proceedings of AAAI*, 2006.

[14] M. Richardson, E. Dominowska, and R. Ragno. Predicting clicks: Estimating the click-through rate for new ads. In *Proceedings of WWW 2007*, 2007.

[15] I. Soboroff. Dynamic test collections: measuring search effectiveness on the live web. In *Proceedings of SIGIR*, pages 276–283, 2006.

[16] I. Soboroff, C. Nicholas, and P. Cahan. Ranking Retrieval Systems without Relevance Judgments. In *Proceedings of SIGIR*, pages 66–73, 2001.

[17] L. Wasserman. *All of Nonparametric Statistics*. Springer, 2006.

[18] S. N. Wood. Thin plate regression splines. *Journal of the Royal Statistical Society: Series B (Statistical Methodology)*, 65(1):95–114, 2003.

[19] T. W. Yee and C. J. Wild. Vector generalized additive models. *Journal of the Royal Statistical Society, Series B (Methodological)*, 58(3):481–493, 1996.

[20] J. Zobel. How Reliable are the Results of Large-Scale Information Retrieval Experiments? In *Proceedings of SIGIR*, pages 307–314, 1998.

